# Chemosensory processing in a spiking model of the olfactory bulb: chemotopic convergence and center surround inhibition

**Baranidharan Raman and Ricardo Gutierrez-Osuna**
Department of Computer Science
Texas A&M University
College Station, TX 77840
*{barani,rgutier}@cs.tamu.edu*

## Abstract

This paper presents a neuromorphic model of two olfactory signal-processing primitives: chemotopic convergence of olfactory receptor neurons, and center on-off surround lateral inhibition in the olfactory bulb. A self-organizing model of receptor convergence onto glomeruli is used to generate a spatially organized map, an olfactory image. This map serves as input to a lattice of spiking neurons with lateral connections. The dynamics of this recurrent network transforms the initial olfactory image into a spatio-temporal pattern that evolves and stabilizes into odor- and intensity-coding attractors. The model is validated using experimental data from an array of temperature-modulated gas sensors. Our results are consistent with recent neurobiological findings on the antennal lobe of the honeybee and the locust.

## 1 Introduction

An artificial olfactory system comprises of an array of cross-selective chemical sensors followed by a pattern recognition engine. An elegant alternative for the processing of sensor-array signals, normally performed with statistical pattern recognition techniques [1], involves adopting solutions from the biological olfactory system. The use of neuromorphic approaches provides an opportunity for formulating new computational problems in machine olfaction, including mixture segmentation, background suppression, olfactory habituation, and odor-memory associations.

A biologically inspired approach to machine olfaction involves (1) identifying key signal processing primitives in the olfactory pathway, (2) adapting these primitives to account for the unique properties of chemical sensor signals, and (3) applying the models to solving specific computational problems.

The biological olfactory pathway can be divided into three general stages: (i) olfactory epithelium, where primary reception takes place, (ii) olfactory bulb (OB), where the bulk of signal processing is performed and, (iii) olfactory cortex, where odor associations are stored. A review of literature on olfactory signal processing reveals six key primitives in the olfactory pathway that can be adapted for use in machine olfaction. These primitives are: (a) chemical transduction into a combinatorial code by a large population of olfactory receptor neurons (ORN), (b) chemotopic convergence of ORN axons onto glomeruli (GL), (c) logarithmic compression through lateral inhibition at the GL level by periglomerular interneurons, (d) contrast enhancement through lateral inhibition of mitral (M) projection neurons by granule interneurons, (e) storage and association of odor memories in the piriform cortex, and (f) bulbar modulation through cortical feedback [2, 3].

This article presents a model that captures the first three abovementioned primitives: population coding, chemotopic convergence and contrast enhancement. The model operates as follows. First, a large population of cross-selective pseudo-sensors is generated from an array of metal-oxide (MOS) gas sensors by means of temperature modulation. Next, a self-organizing model of convergence is used to cluster these pseudo-sensors according to their relative selectivity. This clustering generates an initial spatial odor map at the GL layer. Finally, a lattice of spiking neurons with center on-off surround lateral connections is used to transform the GL map into identity- and intensity-specific attractors.

The model is validated using a database of temperature-modulated sensor patterns from three analytes at three concentration levels. The model is shown to address the first problem in biologically-inspired machine olfaction: intensity and identity coding of a chemical stimulus in a manner consistent with neurobiology [4, 5].

## 2   Modeling chemotopic convergence

The projection of sensory signals onto the olfactory bulb is organized such that ORNs expressing the same receptor gene converge onto one or a few GLs [3]. This convergence transforms the initial combinatorial code into an organized spatial pattern (i.e., an olfactory image). In addition, massive convergence improves the signal to noise ratio by integrating signals from multiple receptor neurons [6]. When incorporating this principle into machine olfaction, a fundamental difference between the artificial and biological counterparts must be overcome: the input dimensionality at the receptor/sensor level. The biological olfactory system employs a large population of ORNs (over 100 million in humans, replicated from 1,000 primary receptor types), whereas its artificial analogue uses a few chemical sensors (commonly one replica of up to 32 different sensor types).

To bridge this gap, we employ a sensor excitation technique known as temperature modulation [7]. MOS sensors are conventionally driven in an isothermal fashion by maintaining a constant temperature. However, the selectivity of these devices is a function of the operating temperature. Thus, capturing the sensor response at multiple temperatures generates a wealth of additional information as compared to the isothermal mode of operation. If the temperature is modulated slow enough (e.g., mHz), the behavior of the sensor at each point in the temperature cycle can then be treated as a pseudo-sensor, and thus used to simulate a large population of cross-selective ORNs (refer to Figure 1(a)).

To model chemotopic convergence, these temperature-modulated pseudo-sensors (referred to as ORNs in what follows) must be clustered according to their

selectivity [8]. As a first approximation, each ORN can be modeled by an affinity vector [9] consisting of the responses across a set of C analytes:

$$\vec{K}_i = \left[ K_i^1, K_i^2, ..., K_i^C \right] \tag{1}$$

where $K_i^a$ is the response of the $i^{th}$ ORN to analyte $a$. The *selectivity* of this ORN is then defined by the orientation of the affinity vector $\vec{K}_i$.

A close look at the OB also shows that neighboring GLs respond to similar odors [10]. Therefore, we model the ORN-GL projection with a Kohonen self-organizing map (SOM) [11]. In our model, the SOM is trained to model the distribution of ORNs in chemical sensitivity space, defined by the affinity vector $\vec{K}_i$. Once the training of the SOM is completed, each ORN is assigned to the closest SOM node (a simulated GL) in affinity space, thereby forming a convergence map. The response of each GL can then be computed as

$$G_j^a = \sigma\left( \sum_{i=1}^{N} W_{ij} \cdot ORN_i^a \right) \tag{2}$$

where $ORN_i^a$ is the response of pseudo-sensor $i$ to analyte $a$, $W_{ij}=1$ if pseudo-sensor $i$ converges to GL $j$ and zero otherwise, and $\sigma(\cdot)$ is a squashing sigmoidal function that models saturation.

This convergence model works well under the assumption that the different sensory inputs are reasonably uncorrelated. Unfortunately, most gas sensors are extremely collinear. As a result, this convergence model degenerates into a few dominant GLs that capture most of the sensory activity, and a large number of dormant GLs that do not receive any projections. To address this issue, we employ a form of competition known as conscience learning [12], which incorporates a habituation mechanism to prevent certain SOM nodes from dominating the competition. In this scheme, the fraction of times that a particular SOM node wins the competition is used as a bias to favor non-winning nodes. This results in a spreading of the ORN projections to neighboring units and, therefore, significantly reduces the number of dormant units.

We measure the performance of the convergence mapping with the entropy across the lattice, $H = -\sum P_i \log P_i$, where $P_i$ is the fraction of ORNs that project to SOM node $i$ [13]. To compare Kohonen and conscience learning, we built convergence mappings with 3,000 pseudo-sensors and 400 GL units (refer to section 4 for details). The theoretical maximum of the entropy for this network, which corresponds to a uniform distribution, is 8.6439. When trained with Kohonen's algorithm, the entropy of the SOM is 7.3555. With conscience learning, the entropy increases to 8.2280. Thus, conscience is an effective mechanism to improve the spreading of ORN projections across the GL lattice.

## 3   Modeling the olfactory bulb network

Mitral cells, which synapse ORNs at the GL level, transform the initial olfactory image into a spatio-temporal code by means of lateral inhibition. Two roles have been suggested for this lateral inhibition: (a) sharpening of the molecular tuning range of individual M cells with respect to that of their corresponding ORNs [10], and (b) global redistribution of activity, such that the bulb-wide representation of an odorant, rather than the individual tuning ranges, becomes specific and concise over time [3]. More recently, center on-off surround inhibitory connections have been found in the OB [14]. These circuits have been suggested to perform pattern normalization, noise reduction and contrast enhancement of the spatial patterns.

We model each M cell using a leaky integrate-and-fire spiking neuron [15]. The input current $I(t)$ and change in membrane potential $u(t)$ of a neuron are given by:

$$I(t) = \frac{u(t)}{R} + C\frac{du}{dt}$$

$$\tau\frac{du}{dt} = -u(t) + R \cdot I(t) \quad [\tau = RC] \tag{3}$$

Each M cell receives current $I_{input}$ from ORNs and current $I_{lateral}$ from lateral connections with other M cells:

$$I_{input}(j) = \sum_i W_{ij} \cdot ORN_i$$

$$I_{lateral}(j,t) = \sum_k L_{kj} \cdot \alpha(k,t-1) \tag{4}$$

where $W_{ij}$ indicates the presence/absence of a synapse between $ORN_i$ and $M_j$, as determined by the chemotopic mapping, $L_{kj}$ is the efficacy of the lateral connection between $M_k$ and $M_j$, and $\alpha(k,t-1)$ is the post-synaptic current generated by a spike at $M_k$:

$$\alpha(k,t-1) = -g(k,t-1) \cdot [u(j,t-1)_+ - E_{syn}] \tag{5}$$

$g(k,t-1)$ is the conductance of the synapse between $M_k$ and $M_j$ at time $t-1$, $u(j,t-1)$ is the membrane potential of $M_j$ at time $t-1$ and the + subscript indicates this value becomes zero if negative, and $E_{syn}$ is the reverse synaptic potential. The change in conductance of post-synaptic membrane is:

$$\dot{g}(k,t) = \frac{dg(k,t)}{dt} = \frac{-g(k,t)}{\tau_{syn}} + z(k,t)$$

$$\dot{z}(k,t) = \frac{dz(k,t)}{dt} = \frac{-z(k,t)}{\tau_{syn}} + g_{norm} \cdot spk(k,t) \tag{6}$$

where $z(.)$ and $g(.)$ are low pass filters of the form $exp(-t/\tau_{syn})$ and $t \cdot \exp(-t/\tau_{syn})$, respectively, $\tau_{syn}$ is the synaptic time constant, $g_{norm}$ is a normalization constant, and $spk(j,t)$ marks the occurrence of a spike in neuron $i$ at time $t$:

$$spk(j,t) = \begin{cases} 1 & u(j,t) = V_{spike} \\ 0 & u(j,t) \neq V_{spike} \end{cases} \tag{7}$$

Combining equations (3) and (4), the membrane potential can be expressed as:

$$\dot{u}(j,t) = \frac{du(j,t)}{dt} = \frac{-u(j,t)}{RC} + \frac{I_{lateral}(j,t)}{C} + \frac{I_{input}(j)}{C}$$

$$u(j,t) = \begin{cases} u(j,t-1) + \dot{u}(j,t-1) \cdot dt & u(j,t) < V_{threshold} \\ V_{spike} & u(j,t) \geq V_{threshold} \end{cases} \tag{8}$$

When the membrane potential reaches $V_{threshold}$, a spike is generated, and the membrane potential is reset to $V_{rest}$. Any further inputs to the neuron are ignored during the subsequent refractory period.

Following [14], lateral interactions are modeled with a center on-off surround matrix $L_{ij}$. Each M cell makes excitatory synapses to nearby M cells $(d<d_e)$, where $d$ is the Manhattan distance measured in the lattice, and inhibitory synapses with

distant M cells $(d_e<d<d_i)$ through granule cells (implicit in our model). Excitatory synapses are assigned uniform random weights between [0, 0.1]. Inhibitory synapses are assigned negative weights in the same interval. Model parameters are summarized in Table 1.

Table 1. Parameters of the OB spiking neuron lattice

| Parameter | Value | Parameter | Value |
|---|---|---|---|
| Peak synaptic conductance ($G_{peak}$) | 0.01 | Synaptic time constants ($\tau_{syn}$) | 10 ms |
| Capacitance (C) | 1 nF | Total simulation time ($t_{tot}$) | 500 ms |
| Resistance (R) | 10 MOhm | Integration time step (dt) | 1 ms |
| Spike voltage ($V_{spike}$) | 70 mV | Refractory period ($t_{ref}$) | 3 ms |
| Threshold voltage ($V_{threshold}$) | 5 mV | Number of mitral cells (N) | 400 |
| Synapse Reverse potential ($E_{syn}$) | 70 mV | Normalization constant ($g_{norm}$) | 0.0027 |
| Excitatory distance ($d_e$) | $d < \dfrac{1}{6}\sqrt{N}$ | Inhibitory distance ($d_i$) | $\dfrac{1}{6}\sqrt{N} < d < \dfrac{2}{6}\sqrt{N}$ |

## 4  Results

The proposed model is validated on an experimental dataset containing gas sensor signals for three analytes: acetone (A), isopropyl alcohol (B) and ammonia (C), at three different concentration levels per analyte. Two Figaro MOS sensors (TGS 2600, TGS 2620) were temperature modulated using a sinusoidal heater voltage (0-7 V; 2.5min period; 10Hz sampling frequency). The response of the two sensors to the three analytes at the three concentration levels is shown in Figure 1(a). This response was used to generate a population of 3,000 ORNs, which were then mapped onto a GL layer with 400 units arranged as a 20×20 lattice**.**

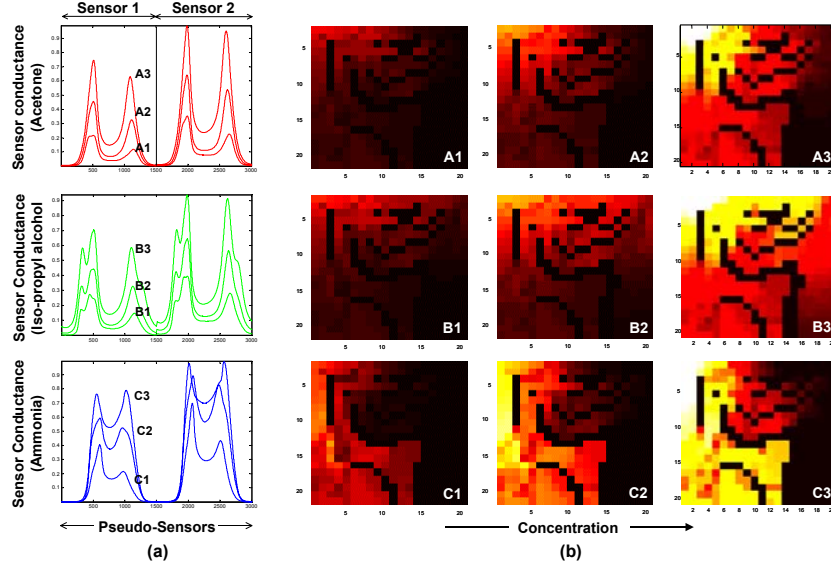

Figure 1. (a) Temperature modulated response to the three analytes (A,B,C) at three concentrations (A3: highest concentration of A), and (b) initial GL maps.

The sensor response to the highest concentration of each analyte was used to generate the SOM convergence map. Figure 1(b) shows the initial odor map of the three analytes following conscience learning of the SOM. These olfactory images show that the identity of the stimulus is encoded by the spatial pattern across the lattice, whereas the intensity is encoded by the overall amplitude of this pattern.

Analytes A and B, which induce similar responses on the MOS sensors, also lead to very similar GL maps.

The GL maps are input to the lattice of spiking neurons for further processing. As a result of the dynamics induced by the recurrent connections, these initial maps are transformed into a spatio-temporal pattern. Figure 2 shows the projection of membrane potential of the 400 M cells along their first three principal components. Three trajectories are shown per analyte, which correspond to the sensor response to the highest analyte concentration on three separate days of data collection. These results show that the spatio-temporal pattern is robust to the inherent drift of chemical sensors. The trajectories originate close to each other, but slowly migrate and converge into unique odor-specific attractors. It is important to note that these trajectories do not diverge indefinitely, but in fact settle into an attractor, as illustrated by the insets in Figure 2.

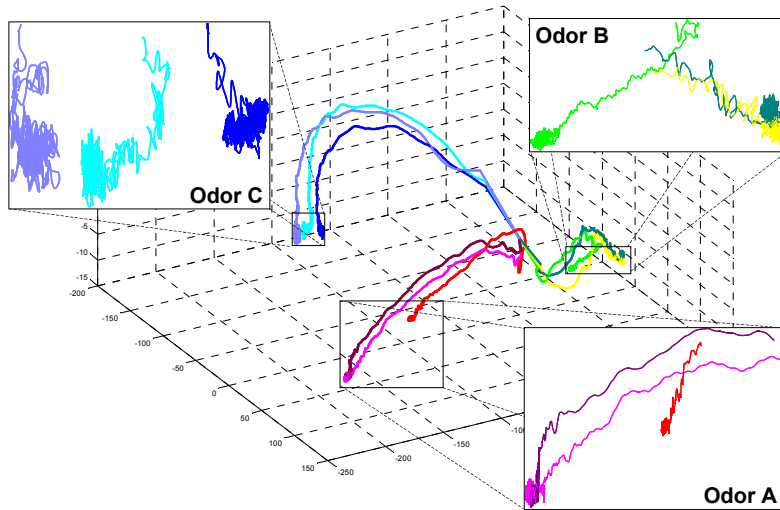

Figure 2. Odor-specific attractors from experimental sensor data. Three trajectories are shown per analyte, corresponding to the sensor response on three separate days. These results show that the attractors are repeatable and robust to sensor drift.

To illustrate the coding of identity and intensity performed by the model, Figure 3 shows the trajectories of the three analytes at three concentrations. The OB network activity evolves to settle into an attractor, where the identity of the stimulus is encoded by the direction of the trajectory relative to the initial position, and the intensity is encoded by the length along the trajectory. This emerging code is also consistent with recent findings in neurobiology, as discussed next.

## 5   Discussion

A recent study of spatio-temporal activity in projection neurons (PN) of the honeybee antennal lobe (analogous to M cells in mammalian OB) reveals evolution and convergence of the network activity into odor-specific attractors [4]. Figure 4(a) shows the projection of the spatio-temporal response of the PNs along their first three principal components. These trajectories begin close to each other, and evolve over time to converge into odor specific regions. These experimental results are consistent with the attractor patterns emerging from our model. Furthermore, an experimental study of odor identity and intensity coding in the locust show

hierarchical groupings of spatio-temporal PN activity according to odor identity, followed by odor intensity [5]. Figure 4(b) illustrates this grouping in the activity of 14 PNs when exposed to three odors at five concentrations. Again, these results closely resemble the grouping of attractors in our model, shown in Figure 3.

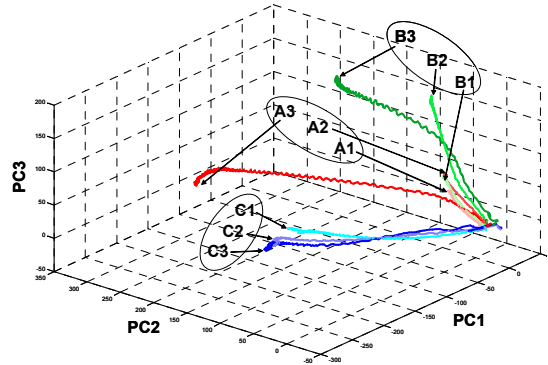

Figure 3. Identity and intensity coding using dynamic attractors.

Previous studies by Pearce et al. [6] using a large population of optical micro-bead chemical sensors have shown that massive convergence of sensory inputs can be used to provide sensory hyperacuity by averaging out uncorrelated noise. In contrast, the focus of our work is on the coding properties induced by chemotopic convergence. Our model produces an initial spatial pattern or olfactory image, whereby odor identity is coded by the spatial activity across the GL lattice, and odor intensity is encoded by the amplitude of this pattern. Hence, the bulk of the identity/intensity coding is performed by this initial convergence primitive.

Subsequent processing by a lattice of spiking neurons introduces time as an additional coding dimension. The initial spatial maps are transformed into a spatio-temporal pattern by means of center on-off surround lateral connections. Excitatory lateral connections allow the model to spread M cell activity, and are responsible for moving the attractors away from their initial coordinates. In contrast, inhibitory connections ensure that these trajectories eventually converge onto an attractor, rather than diverge indefinitely. It is the interplay between excitatory and inhibitory connections that allows the model to enhance the initial coding produced by the chemotopic convergence mapping.

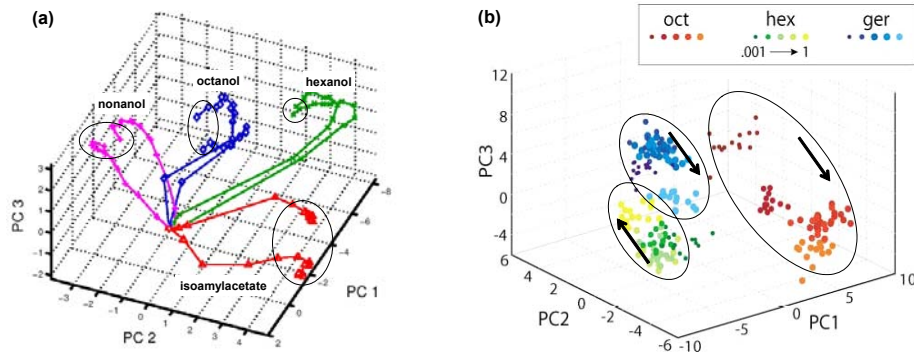

Figure 4. (a) Odor trajectories formed by spatio-temporal activity in the honeybee AL (adapted from [4]). (b) Identity and intensity clustering of spatio-temporal activity in the locust AL (adapted from [5]; arrows indicate the direction of increasing concentration).

At present, our model employs a center on-off surround kernel that is constant throughout the lattice. Further improvements can be achieved through adaptation of these lateral connections by means of Hebbian and anti-Hebbian learning. These extensions will allow us to investigate additional computational functions (e.g., pattern completion, orthogonalization, coding of mixtures) in the processing of information from chemosensor arrays.

## Acknowledgments

This material is based upon work supported by the National Science Foundation under CAREER award 9984426/0229598. Takao Yamanaka, Alexandre Perera-Lluna and Agustin Gutierrez-Galvez are gratefully acknowledged for valuable suggestions during the preparation of this manuscript.

## References

[1]    Gutierrez-Osuna, R. (2002) Pattern Analysis for Machine Olfaction: A Review. *IEEE Sensors Journal* 2(3): 189-202.
[2]    Pearce, T. C. (1999) Computational parallels between the biological olfactory pathway and its analogue 'The Electronic Nose': Part I. Biologiacal olfaction. *BioSystems* 41: 43-67.
[3]    Laurent, G. (1999) A Systems Perspective on Early Olfactory Coding. *Science* 286(22): 723-728.
[4]    Galán, R. F.,Sachse, S., Galizia, C.G., & Herz, A.V. (2003) Odor-driven attractor dynamics in the antennal lobe allow for simple and rapid olfactory pattern classification. *Neural Computation* 16(5): 999-1012.
[5]    Stopfer, M., Jayaraman, V., & Laurent, G. (2003) Intensity versus Identity Coding in an Olfactory System. *Neuron* 39: 991-1004.
[6]    Pearce, T.C., Verschure, P.F.M.J., White, J., & Kauer, J. S. (2001) Robust Stimulus Encoding in Olfactory Processing: Hyperacuity and Efficient Signal Transmission. In S. Wermter, J. Austin and D. Willshaw (Eds.), *Emergent Neural Computation Architectures Based on Neuroscience*. pp. 461-479. Springer-Verlag.
[7]    Lee. A. P., & Reedy, B. J. (1999) Temperature modulation in semiconductor gas sensing. *Sensors and Actuators B* 60: 35-42.
[8]    Vassar, R., Chao, S.K., Sitcheran, R., Nunez, J. M., Vosshall, L.B., & Axel, A. (1994) Topographic Organization of Sensory Projections to the Olfactory Bulb. *Cell* 79(6): 981-991.
[9]    Gutierrez-Osuna, R. (2002) A Self-organizing Model of Chemotopic Convergence for Olfactory Coding. In *Proceedings of the 2$^{nd}$ EMBS-BMES Conference*, pp. 23-26. Texas.
[10]   Mori, K., Nagao, H., & Yoshihara, Y. (1999) The Olfactory Bulb: Coding and Processing of Odor molecule information. *Science* 286: 711-715.
[11]   Kohonen, T. (1982) Self-organized formation of topologically correct feature maps. *Biological Cybernetics* 43: 59-69.
[12]   DeSieno, D. (1988) Adding conscience to competitive learning. In *Proceedings of International Conference on Neural Networks (ICNN)*, pp. 117-124. Piscataway, NJ.
[13]   Laaksonen, J., Koskela, M., & Oja, E. (2003) Probability interpretation of distributions on SOM surfaces. In *Proceedings of Workshop on Self-Organizing Maps*. Hibikino, Japan.
[14]   Aungst et al. (2003) Center-surround inhibition among olfactory bulb glomeruli. *Nature* 26: 623- 629.
[15]   Gerstner, W., & Kistler, W. (2002) *Spiking Neuron Models: Single Neurons, Populations, Plasticity*. Cambridge, University Press.
